# Discriminative Log-Linear Grammars with Latent Variables

**Slav Petrov and Dan Klein**
Computer Science Department, EECS Division
University of California at Berkeley, Berkeley, CA, 94720
{petrov, klein}@cs.berkeley.edu

## Abstract

We demonstrate that log-linear grammars with latent variables can be practically trained using discriminative methods. Central to efficient discriminative training is a hierarchical pruning procedure which allows feature expectations to be efficiently approximated in a gradient-based procedure. We compare $L_1$ and $L_2$ regularization and show that $L_1$ regularization is superior, requiring fewer iterations to converge, and yielding sparser solutions. On full-scale treebank parsing experiments, the discriminative latent models outperform both the comparable generative latent models as well as the discriminative non-latent baselines.

## 1  Introduction

In recent years, latent annotation of PCFG has been shown to perform as well as or better than standard lexicalized methods for treebank parsing [1, 2]. In the latent annotation scenario, we imagine that the observed treebank is a coarse trace of a finer, unobserved grammar. For example, the single treebank category NP (noun phrase) may be better modeled by several finer categories representing subject NPs, object NPs, and so on. At the same time, discriminative methods have consistently provided advantages over their generative counterparts, including less restriction on features and greater accuracy [3, 4, 5]. In this work, we therefore investigate discriminative learning of latent PCFGs, hoping to gain the best from both lines of work.

Discriminative methods for parsing are not new. However, most discriminative methods, at least those which globally trade off feature weights, require repeated parsing of the training set, which is generally impractical. Previous work on end-to-end discriminative parsing has therefore resorted to "toy setups," considering only sentences of length 15 [6, 7, 8] or extremely small corpora [9]. To get the benefits of discriminative methods, it has therefore become common practice to extract n-best candidate lists from a generative parser and then use a discriminative component to rerank this list. In such an approach, repeated parsing of the training set can be avoided because the discriminative component only needs to select the best tree from a fixed candidate list. While most state-of-the-art parsing systems apply this hybrid approach [10, 11, 12], it has the limitation that the candidate list often does not contain the correct parse tree. For example 41% of the correct parses were not in the candidate pool of $\approx$30-best parses in [10].

In this paper we present a hierarchical pruning procedure that exploits the structure of the model and allows feature expectations to be efficiently approximated, making discriminative training of full-scale grammars practical. We present a gradient-based procedure for training a discriminative grammar on the entire WSJ section of the Penn Treebank (roughly 40,000 sentences containing 1 million words). We then compare $L_1$ and $L_2$ regularization and show that $L_1$ regularization is superior, requiring fewer iterations to converge and yielding sparser solutions. Independent of the regularization, discriminative grammars significantly outperform their generative counterparts in our experiments.

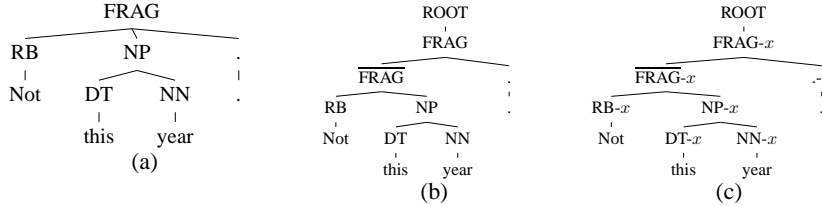

Figure 1: (a) The original tree. (b) The (binarized) X-bar tree. (c) The annotated tree.

## 2   Grammars with latent annotations

Context-free grammars (CFGs) underlie most high-performance parsers in one way or another [13, 12, 14]. However, a CFG which simply takes the empirical productions and probabilities off of a treebank does not perform well. This naive grammar is a poor one because its context-freedom assumptions are too strong in some places and too weak in others. Therefore, a variety of techniques have been developed to both enrich and generalize the naive grammar. Recently an automatic state-splitting approach was shown to produce state-of-the art performance [2, 14]. We extend this line of work by investigating discriminative estimation techniques for automatically refined grammars.

We consider grammars that are automatically derived from a raw treebank. Our experiments are based on a completely unsplit X-bar grammar, obtained directly from the Penn Treebank by the binarization procedure shown in Figure 1. For each local tree rooted at an evaluation category $X$, we introduce a cascade of new nodes labeled $\overline{X}$ so that each has two children in a right branching fashion. Each node is then refined with a latent variable, splitting each observed category into $k$ unobserved subcategories. We refer to trees over unsplit categories as *parse trees* and trees over split categories as *derivations*.

Our log-linear grammars are parametrized by a vector $\theta$ which is indexed by productions $X \to \gamma$. The conditional probability of a derivation tree $t$ given a sentence $w$ can be written as:

$$P_\theta(t|w) = \frac{1}{Z(\theta,w)} \prod_{X \to \gamma \in t} e^{\theta_{X \to \gamma}} = \frac{1}{Z(\theta,w)} e^{\theta^\mathsf{T} f(t)} \tag{1}$$

where $Z(\theta,w)$ is the partition function and $f(t)$ is a vector indicating how many times each production occurs in the derivation $t$. The inside/outside algorithm [15] gives us an efficient way of summing over an exponential number of derivations. Given a sentence $w$ spanning the words $w^1, w^2, \ldots, w^n = w^{1:n}$, the inside and outside scores of a (split) category $A$ spanning $(i,j)$ are computed by summing over all possible children $B$ and $C$ spanning $(i,k)$ and $(k,j)$ respectively:[1]

$$S_{\text{IN}}(A,i,j) = \sum_{A \to BC} \sum_{i < k < j} \phi_{A \to BC} \times S_{\text{IN}}(B,i,k) \times S_{\text{IN}}(C,k,j)$$

$$S_{\text{OUT}}(A,i,j) = \sum_{B \to CA} \sum_{1 \le k < i} \phi_{B \to CA} \times S_{\text{OUT}}(B,k,j) \times S_{\text{IN}}(C,k,i) +$$

$$\sum_{B \to AC} \sum_{j < k \le n} \phi_{B \to AC} \times S_{\text{OUT}}(B,i,k) \times S_{\text{IN}}(C,j,k), \tag{2}$$

where we use $\phi_{A \to BC} = e^{\theta_{A \to BC}}$. In the generative case these scores correspond to the inside and outside probabilities $S_{\text{IN}}(A,i,j) = P_{\text{IN}}(A,i,j) \overset{\text{def}}{=} P(w^{i:j}|A)$ and $S_{\text{OUT}}(A,i,j) = P_{\text{OUT}}(A,i,j) \overset{\text{def}}{=} P(w^{1:i}Aw^{j:n})$ [15]. The scores lack this probabilistic interpretation in the discriminative case, but they can nonetheless be normalized in the same way as probabilities to produce the expected counts of productions needed at training time. The posterior probability of a production $A \to BC$ spanning $(i,j)$ with split point $k$ in a sentence is easily expressed as:

$$\langle A \to BC, i, j, k \rangle \quad \propto \quad S_{\text{OUT}}(A,i,j) \times \phi_{A \to BC} \times S_{\text{IN}}(B,i,k) \times S_{\text{IN}}(C,k,j) \tag{3}$$

To obtain a grammar from the training trees, we want to learn a set of grammar parameters $\theta$ on latent annotations despite the fact that the original trees lack the latent annotations. We will consider

generative grammars, where the parameters $\theta$ are set to maximize the joint likelihood of the training sentences and their parse trees, and discriminative grammars, where the parameters $\theta$ are set to maximize the likelihood of the correct parse tree (vs. all possible trees) given a sentence. Previous work on automatic grammar refinement has focused on different estimation techniques for learning generative grammars with latent labels (training with basic EM [1], an EM-based split and merge approach [2], a non-parametric variational approach [16]). In the following, we review how generative grammars are learned and present an algorithm for estimating discriminative grammars with latent variables.

## 2.1 Generative Grammars

Generative grammars with latent variables can be seen as tree structured hidden Markov models. A simple EM algorithm [1] allows us to learn parameters for generative grammars which maximize the log joint likelihood of the training sentences $w$ and parse trees $T$:

$$\mathcal{L}_{joint}(\theta) = \log \prod_i P_\theta(w_i, T_i) = \log \prod_i \sum_{t:T_i} P_\theta(w_i, t), \tag{4}$$

where $t$ are derivations (over split categories) corresponding to the observed parse tree (over unsplit categories). In the E-Step we compute inside/outside scores over the set of derivations corresponding to the observed gold tree by restricting the sums in Eqn. 2 to produce only such derivations. [2] We then use Eqn. 3 to compute expectations which are normalized in the M-Step to update the production probabilities $\phi_{X\to\gamma} = e^{\theta_{X\to\gamma}}$ to their maximum likelihood estimates:

$$\phi_{X\to\gamma} = \frac{\sum_T \mathbb{E}_\theta[f_{X\to\gamma}(t)|T]}{\sum_{\gamma'}\sum_T \mathbb{E}_\theta[f_{X\to\gamma'}(t)|T]} \tag{5}$$

Here, $\mathbb{E}_\theta[f_{X\to\gamma}(t)|T]$ denotes the expected count of the production (or feature) $X \to \gamma$ with respect to $P_\theta$ in the set of derivations t, which are consistent with the observed parse tree $T$. Similarly, we will write $\mathbb{E}_\theta[f_{X\to\gamma}(t)|w]$ for the expectation over all derivations of the sentence $w$.

Our generative grammars with latent variables are probabilistic context-free grammars (CFGs), where $\sum_{\gamma'} \phi_{X\to\gamma'} = 1$ and $Z(\theta) = 1$. Note, however, that this normalization constraint poses no restriction on the model class, as probabilistic and weighted CFGs are equivalent [18].

## 2.2 Discriminative Grammars

Discriminative grammars with latent variables can be seen as conditional random fields [4] over trees. For discriminative grammars, we maximize the log conditional likelihood:

$$\mathcal{L}_{cond}(\theta) = \log \prod_i P_\theta(T_i|w_i) = \log \prod_i \sum_{t:T_i} \frac{e^{\theta^\top f(t)}}{Z(\theta, w_i)} \tag{6}$$

We directly optimize this non-convex objective function using a numerical gradient based method (LBFGS [19] in our implementation).[3] Fitting the log-linear model involves the following derivatives:

$$\frac{\partial \mathcal{L}_{cond}(\theta)}{\partial \theta_{X\to\gamma}} = \sum_i \left( \mathbb{E}_\theta[f_{X\to\gamma}(t)|T_i] - \mathbb{E}_\theta[f_{X\to\gamma}(t)|w_i] \right), \tag{7}$$

where the first term is the expected count of a production in derivations corresponding to the correct parse tree and the second term is the expected count of the production in all parses.

The challenge in estimating discriminative grammars is that the computation of some quantities requires repeatedly taking expectations over all parses of all sentences in the training set. We will discuss ways to make their computation on large data sets practical in the next section.

# 3 Efficient Discriminative Estimation

Computing the partition function in Eqn. 6 requires parsing of the entire training corpus. Even with recent advances in parsing efficiency and fast CPUs, parsing the entire corpus repeatedly remains prohibitive. Fast parsers like [12, 14] can parse several sentences per second, but parsing the 40,000 training sentences still requires more than 5 hours on a fast machine. Even in a parallel implementation, parsing the training corpus several hundred times, as necessary for discriminative training, would and, in fact, did in the case of maximum margin training [6], require weeks. Generally speaking, there are two ways of speeding up the training process: reducing the total number of training iterations and reducing the time required per iteration.

## 3.1 Hierarchical Estimation

The number of training iterations can be reduced by training models of increasing complexity in a hierarchical fashion. For example in mixture modeling [20] and machine translation [21], a sequence of increasingly more complex models is constructed and each model is initialized with its (simpler) predecessor. In our case, we begin with the unsplit X-Bar grammar and iteratively split each category in two and re-train the grammar. In each iteration, we initialize with the results of the smaller grammar, splitting each annotation category in two and adding a small amount of randomness to break symmetry. In addition to reducing the number of training iterations, hierarchical training has been shown to lead to better parameter estimates [2]. However, even with hierarchical training, large-scale discriminative training will remain impractical, unless we can reduce the time required to parse the training corpus.

## 3.2 Feature-Count Approximation

High-performance parsers have employed coarse-to-fine pruning schemes, where the sentence is rapidly pre-parsed with increasingly more complex grammars [22, 14]. Any constituent with sufficiently low posterior probability triggers the pruning of its refined variants in subsequent passes. While this method has no theoretical guarantees, it has been empirically shown to lead to a 100-fold speed-up without producing search errors [14].

Instead of parsing each sentence exhaustively with the most complex grammar in each iteration, we can approximate the expected feature counts by parsing in a hierarchical coarse-to-fine scheme. We start by parsing exhaustively with the X-Bar grammar and then prune constituents with low posterior probability ($e^{-10}$ in our experiments).[4] We then continue to parse with the next more refined grammar, skipping over constituents whose less refined predecessor has been pruned. After parsing with the most refined grammar, we extract expected counts from the final (sparse) chart. The expected counts will be approximations because many small counts have been set to zero by the pruning procedure.

Even though this procedure speeds-up each training iteration tremendously, training remains prohibitively slow. We can make repeated parsing of the same sentences significantly more efficient by *caching* the pruning history from one training iteration to the next. Instead of computing each stage in the coarse-to-fine scheme for every pass, we can compute it once when we start training a grammar and update only the final, most refined scores in every iteration. Cached pruning has the positive side effect of constraining subcategories to refine their predecessors, so that we do not need to worry about issues like subcategory drift and projections [14].

As only extremely unlikely items are removed from the chart, pruning has virtually no effect on the conditional likelihood. Pruning more aggressively leads to a training procedure reminiscent of *contrastive estimation* [23], where the denominator is restricted to a neighborhood of the correct parse tree (rather than containing all possible parse trees). In our experiments, pruning more aggressively did not hurt performance for grammars with few subcategories, but limited the performance of grammars with many subcategories.

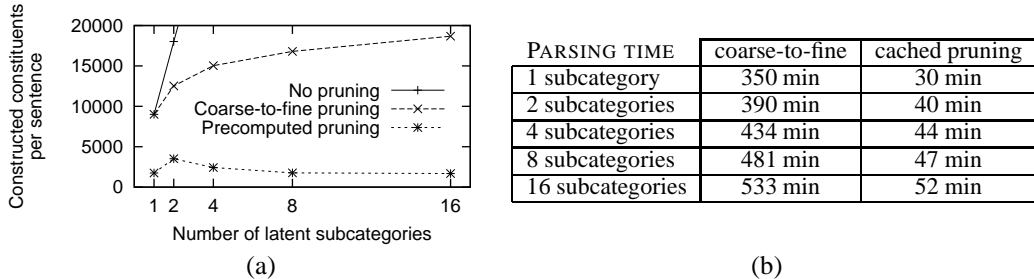

| PARSING TIME | coarse-to-fine | cached pruning |
|---|---|---|
| 1 subcategory | 350 min | 30 min |
| 2 subcategories | 390 min | 40 min |
| 4 subcategories | 434 min | 44 min |
| 8 subcategories | 481 min | 47 min |
| 16 subcategories | 533 min | 52 min |

(a)                                                     (b)

Figure 2: Average number of constructed constituents per sentence (a) and time to parse the training corpus for different pruning regimes and grammar sizes (b).

## 4  Results

We ran our experiments on the Wall Street Journal (WSJ) portion of the English Penn Treebank using the standard setup: we trained on sections 2 to 21. Section 22 was used as development set for intermediate results. All of section 23 was reserved for the final test. We used the EVALB parseval reference implementation for scoring. We will report $F_1$-scores[5] and exact match percentages. For the final test, we selected the grammar that performed best on the development set.

For our lexicon, we used a simple approach where rare words (seen five times or less during training) are replaced by one of 50 unknown word tokens based on a small number of word-form features. To parse new sentences with a grammar, we compute the posterior distribution over productions at each span and extract the tree with the maximum expected number of correct productions [14].

### 4.1  Efficiency

The average number of constituents that are constructed while parsing a sentence is a good indicator for the efficiency of our cached pruning scheme.[6] Figure 2(a) shows the average number of chart items that are constructed per sentence. Coarse-to-fine pruning refers to hierarchical pruning without caching [14] and while it is better than no-pruning, it still constructs a large number of constituents for heavily refined grammars. In contrast, with cached pruning the number of constructed chart items stays roughly constant (or even decreases) when the number of subcategories increases. The reduced number of constructed constituents results in a 10-fold reduction of parsing time, see Figure 2(b), and makes discriminative training on a large scale corpus computationally feasible.

We found that roughly 100-150 training iterations were needed for LBFGS to converge after each split. Distributing the training over several machines is straightforward as each sentence can be parsed independently of all other sentences. Starting from an unsplit X-Bar grammar we were able to hierarchically train a 16 substate grammar in three days using eight CPUs in parallel.[7]

It should be also noted that we can expedite training further by training in an interleaved mode, where after splitting a grammar we first run generative training for some time (which is very fast) and then use the resulting grammar to initialize the discriminative training. In such a training regime, we only needed around 50 iterations of discriminative training until convergence, significantly speeding up the training, while maintaining the same final performance.

### 4.2  Regularization

Regularization is often necessary to prevent discriminative models from overfitting on the training set. Surprisingly enough, we found that no regularization was necessary when training on the entire training set, even in the presence of an abundance of features. During development we trained on subsets of the training corpus and found that regularization was crucial for preventing overfit-

|  | EXACT MATCH | | $F_1$-SCORE | |
| --- | --- | --- | --- | --- |
|  | generative | discriminative | generative | discriminative |
| 1 subcategory | 7.6 | 7.8 | 64.8 | 67.3 |
| 2 subcategories | 14.6 | 20.1 | 76.4 | 80.8 |
| 4 subcategories | 24.6 | 31.3 | 83.7 | 85.6 |
| 8 subcategories | 31.4 | 37.0 | 86.6 | 87.8 |
| 16 subcategories | 35.8 | **39.4** | 88.7 | **89.3** |

Table 1: Discriminative training is superior to generative training for exact match and for $F_1$-score.

|  | $L_1$ regularization | | | | $L_2$ regularization | | | |
| --- | --- | --- | --- | --- | --- | --- | --- | --- |
|  | $F_1$-score | Exact | # Feat. | # Iter. | $F_1$-score | Exact | # Feat. | # Iter. |
| 1 subcategory | 67.3 | 7.8 | 23 K | 44 | 67.4 | 7.9 | 35 K | 67 |
| 2 subcategories | 80.8 | 20.1 | 74 K | 108 | 80.3 | 19.5 | 123 K | 132 |
| 4 subcategories | 85.6 | 31.3 | 147 K | 99 | 85.7 | 31.5 | 547 K | 148 |
| 8 subcategories | 87.8 | 37.0 | 318 K | 82 | 87.6 | 36.9 | 2,983 K | 111 |
| 16 subcategories | 89.3 | 39.4 | 698 K | 75 | 89.1 | 38.7 | 11,489 K | 102 |

Table 2: $L_1$ regularization produces sparser solutions and requires fewer training iterations than $L_2$ regularization.

ting. This result is in accordance with [16] where a variational Bayesian approach was found to be beneficial for small training sets but performed on par with EM for large amounts of training data.

Regularization is achieved by adding a penalty term to the conditional log likelihood function $\mathcal{L}_{cond}(\theta)$. This penalty term is often a weighted norm of the parameter vector and thereby penalizes large parameter values. We investigated $L_1$ and $L_2$ regularization:

$$\mathcal{L}'_{cond}(\theta) = \mathcal{L}_{cond}(\theta) - \frac{1}{2} \sum_{X \to \gamma} \frac{|\theta_{X \to \gamma}|}{\sigma} \qquad \mathcal{L}''_{cond}(\theta) = \mathcal{L}_{cond}(\theta) - \sum_{X \to \gamma} \left( \frac{\theta_{X \to \gamma}}{\sigma} \right)^2 \quad (8)$$

where the regularization parameter $\sigma$ is tuned on a held out set. In the $L_2$ case, the penalty term is a convex and differentiable function of the parameters and hence can be easily intergrated into our training procedure. In the $L_1$ case, however, the penalty term is discontinuous whenever some parameter equals zero. To handle the discontinuinty of the gradient, we used the orthant-wise limited-memory quasi-Newton algorithm of [24].

Table 2 shows that while there is no significant performance difference in models trained with $L_1$ or $L_2$ regularization, there is significant difference in the number of training iterations and the sparsity of the parameter vector. $L_1$ regularization leads to extremely sparse parameter vectors (96% of the parameters are zero in the 16 subcategory case), while no parameter value becomes exactly zero with $L_2$ regularization. It remains to be seen how this sparsity can be exploited, as these zeros become ones when exponentiated in order to be used in the computation of inside and outside scores.

### 4.3 Final Test Set Results

Table 1 shows a comparison of generative and discriminative grammars for different numbers of subcategories. Discriminative training is superior to generative training for exact match as well as for $F_1$-score for all numbers of subcategories. For our largest grammars, we see absolute improvements of 3.63% and 0.61% in exact match and $F_1$ score respectively. The better performance is due to better parameter estimates, as the model classes defined by the generative and discriminative model (probabilistic vs. weighted CFGs) are equivalent [18] and the same feature sets were used in all experiments.

Our final test set parsing $F_1$-score of 88.8/88.3 (40 word sentences/all sentences) is better than most other systems, including basic generative latent variable grammars [1] ($F_1$-score of 86.7/86.1) and even fully lexicalized systems [13] ($F_1$-score of 88.6/88.2), but falls short of the very best systems [12, 14], which achieve accuracies above 90%. However, many of the techniques used in [12, 14] are orthogonal to what was presented here (additional non-local/overlapping features, merging of unnecessary splits) and could be incorporated into the discriminative model.

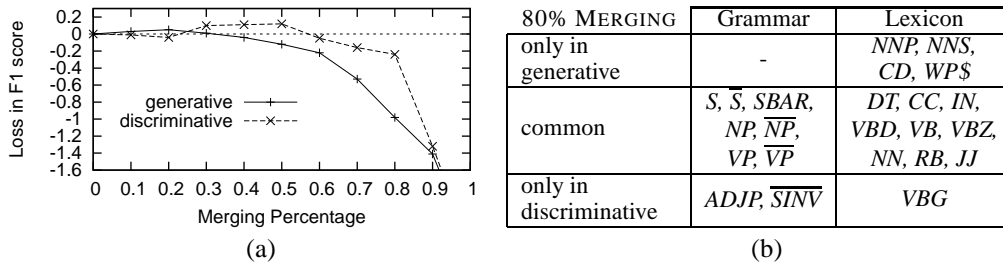

| 80% MERGING | Grammar | Lexicon |
|---|---|---|
| only in generative | - | NNP, NNS, CD, WP$ |
| common | S, $\overline{S}$, SBAR, NP, $\overline{NP}$, VP, $\overline{VP}$ | DT, CC, IN, VBD, VB, VBZ, NN, RB, JJ |
| only in discriminative | ADJP, $\overline{SINV}$ | VBG |

|  |  |
|---|---|
| (a) | (b) |

Figure 3: (a) Loss in $F_1$ score for different amounts of merging. (b) Categories with two subcategories after merging 80% of the subcategories according to the merging criterion in [2].

## 4.4 Analysis

Generatively trained grammars with latent variables have been shown to exhibit many linguistically interpretable phenomena [2]. Space does not permit a thorough exposition, and *post hoc* analysis of learned structures is prone to seeing what one expects, but nonetheless it can be helpful to illustrate the broad patterns that are learned. Not surprisingly, many comparable trends can be observed in generatively and discriminatively trained grammars. For example, the same subdivisions of the determiner category (*DT*) into definite (*the*), indefinite (*a*), demonstrative (*this*) and quantificational (*some*) elements emerge under both training regimes. Another example is the preposition category (*IN*) where subcategories for subordinating conjunctions like (*that*) and different types of proper prepositions are learned. Typically the divisions in the discriminative grammars are much more pronounced, putting the majority of the weight on a few dominant words.

While many similarities can be found, it is especially interesting to examine how generative and discriminative grammars differ. The nominal categories in generative grammars exhibit many clusters of semantic nature (e.g. subcategories for dates, monetary units, capitalized words, etc.). For example, the following two subcategories of the proper noun (*NNP*) category {New, San, Wall} and {York, Francisco, Street} (here represented by the three most likely words) are learned by the generative grammars. These subcategories are very useful for modeling correlations when generating words and many clusters with such semantic patterns appear in the generative grammars. However, these clusters do not interact strongly with disambiguation and are therefore not learned by the discriminative grammars. Similar observations hold for plural proper nouns (*NNPS*), superlative adjectives (*JJS*), and cardinal numbers (*CD*), which are heavily split into semantic subcategories in the generative grammars but are split very little or not at all in the discriminative grammars.

Examining the phrasal splits is much more intricate. We therefore give just one example from grammars with two subcategories, which illustrates the main difference between generative and discriminative grammars. Simple declarative clauses (*S*) are the most common sentences in the Penn Treebank, and in the generative case the most likely expansion of the *ROOT* category is $ROOT \rightarrow S_1$, being chosen 91% of the time. In the discriminative case this production is only the third likeliest with a weight of 13.2. The highest weighted expansion of the *ROOT* in the discriminative grammar is $ROOT \rightarrow SBARQ_1$, with a weight of 46.5, a production that has a probability of 0.3% in the generative grammar. While generative grammars model the empirical distributions of productions in the training set, discriminative grammars maximize the discriminative power of the model. This can for example result in putting the majority of the weight on underrepresented productions.

We applied the merging criterion suggested in [2] to two grammars with two subcategories in order to quantitatively examine how many subcategories are learned. This criterion approximates the loss in joint likelihood incurred from merging two subcategories and we extended it to approximate the loss in conditional likelihood from merging two subcategories at a given node. Figure 3(a) shows the loss in $F_1$-score when the least useful fraction of the subcategories are merged. Our observation that the discriminative grammars learn far fewer clusters are confirmed, as one can merge back 80% of the subcategories at almost no loss in $F_1$ (while one can merge only 50% in the generative case). This suggest that one can learn discriminative grammars which are significantly more compact and accurate than their generative counterparts. Figure 3(b) shows which categories remain split when 80% of the splits are merged. While there is a substantial overlap between the learned splits, one can see that joint likelihood can be better maximized by refining the lexicon, while conditional likelihood is better maximized by refining the grammar.

# 5   Conclusions and Future Work

We have presented a hierarchical pruning procedure that allows efficient discriminative training of log-linear grammars with latent variables. We avoid repeated computation of similar quantities by caching information between training iterations and approximating feature expectations. We presented a direct gradient-based procedure for optimizing the conditional likelihood function which in our experiments on full-scale treebank parsing lead to discriminative latent models which outperform both the comparable generative latent models, as well as the discriminative non-latent baselines. We furthemore investigated different regularization penalties and showed that $L_1$ regularization leads to extremely sparse solutions

While our results are encouraging, this is merely a first investigation into large-scale discriminative training of latent variable grammars and opens the door for many future experiments: discriminative grammars allow the seamless integration of non-local and overlapping features and it will be interesting to see how proven features from reranking systems [10, 11, 12] and other orthogonal improvements like merging and smoothing [2] will perform in an end-to-end discriminative system.

## Footnotes

[1]Although we show only the binary component, of course both binary and unary productions are included.

[2] Since the tree structure is observed this can be done in linear time [17].

[3] Alternatively, maximum conditional likelihood estimation can also be seen as a special case of maximum likelihood estimation, where $P(w)$ is assumed to be the empirical one and not learned. The conditional likelihood optimization can therefore be addressed by an EM algorithm which is similar to the generative case. However, while the E-Step remains the same, the M-Step involves fitting a log-linear model, which requires optimization, unlike the joint case, which can be done analytically using relative frequency estimators. This EM algorithm typically converges to a comparable local maximum as direct optimization of the objective function but requires 3-4 times more iterations.

[4]Even a tighter threshold produced no search errors on a held out set in [14]. We enforce that the gold parse is always reachable.

[5]The harmonic mean of precision $P$ and recall $R$: $\frac{2PR}{P+R}$.

[6]The other main factor determining the parsing time is the grammar size.

[7]Memory limitations prevent us from learning grammars with more subcategories, a problem that could be alleviated by merging back the least usefull splits as in [2].

## References

[1]  T. Matsuzaki, Y. Miyao, and J. Tsujii. Probabilistic CFG with latent annotations. In *ACL '05*, 2005.

[2]  S. Petrov, L. Barrett, R. Thibaux, and D. Klein. Learning accurate, compact, and interpretable tree annotation. In *ACL '06*, 2006.

[3]  A. Y. Ng and M. I. Jordan. On discriminative vs. generative classifiers: A comparison of logistic regression and naive Bayes. In *NIPS '02*, 2002.

[4]  J. Lafferty, A. McCallum, and F. Pereira. Conditional Random Fields: Probabilistic models for segmenting and labeling sequence data. In *ICML '01*, 2001.

[5]  D. Klein and C. Manning. Conditional structure vs conditional estimation in NLP models. In *EMNLP '02*, 2002.

[6]  B. Taskar, D. Klein, M. Collins, D. Koller, and C. Manning. Max-margin parsing. In *EMNLP '04*, 2004.

[7]  J. Henderson. Discriminative training of a neural network statistical parser. In *ACL '04*, 2004.

[8]  J. Turian, B. Wellington, and I. D. Melamed. Scalable discriminative learning for natural language parsing and translation. In *NIPS '07*, 2007.

[9]  M. Johnson. Joint and conditional estimation of tagging and parsing models. In *ACL '01*, 2001.

[10]  M. Collins. Discriminative reranking for natural language parsing. In *ICML '00*, 2000.

[11]  T. Koo and M. Collins. Hidden-variable models for discriminative reranking. In *EMNLP '05*, 2005.

[12]  E. Charniak and M. Johnson. Coarse-to-Fine N-Best Parsing and MaxEnt Discriminative Reranking. In *ACL'05*, 2005.

[13]  M. Collins. *Head-Driven Statistical Models for Natural Language Parsing*. PhD thesis, UPenn., 1999.

[14]  S. Petrov and D. Klein. Improved inference for unlexicalized parsing. In *HLT-NAACL '07*, 2007.

[15]  K. Lari and S. Young. The estimation of stochastic context-free grammars using the inside-outside algorithm. *Computer Speech and Language*, 1990.

[16]  P. Liang, S. Petrov, M. I. Jordan, and D. Klein. The infinite PCFG using hierarchical Dirichlet processes. In *EMNLP '07*, 2007.

[17]  F. Pereira and Y. Schabes. Inside-outside reestimation from partially bracketed corpora. In *ACL*, 1992.

[18]  N. A. Smith and M. Johnson. Weighted and probabilistic context-free grammars are equally expressive. *To appear in Computational Lingusitics*, 2007.

[19]  J. Nocedal and S. J. Wright. *Numerical Optimization*. Springer, 1999.

[20]  N. Ueda, R. Nakano, Z. Ghahramani, and G. E. Hinton. Split and merge EM algorithm for mixture models. *Neural Computation*, 12(9):2109–2128, 2000.

[21]  P. F. Brown, S. A. D. Pietra, V. J. D. Pietra, and R. L. Mercer. The mathematics of statistical machine translation. *Computational Lingusitics*, 19(2), 1993.

[22]  E. Charniak, M. Johnson, D. McClosky, et al. Multi-level coarse-to-fine PCFG Parsing. In *HLT-NAACL '06*, 2006.

[23]  N. A. Smith and J. Eisner. Contrastive estimation: Training log-linear models on unlabeled data. In *ACL '05*, 2005.

[24]  G. Andrew and J. Gao. Scalable training of L1-regularized log-linear models. In *ICML '07*, 2007.

